# Bidirectional Retrieval from Associative Memory

**Friedrich T. Sommer and Günther Palm**
Department of Neural Information Processing
University of Ulm, 89069 Ulm, Germany
{sommer,palm}@informatik.uni-ulm.de

## Abstract

Similarity based fault tolerant retrieval in neural associative memories (NAM) has not lead to wiedespread applications. A drawback of the efficient Willshaw model for sparse patterns [Ste61, WBLH69], is that the high asymptotic information capacity is of little practical use because of high cross talk noise arising in the retrieval for finite sizes. Here a new bidirectional iterative retrieval method for the Willshaw model is presented, called crosswise bidirectional (CB) retrieval, providing enhanced performance. We discuss its asymptotic capacity limit, analyze the first step, and compare it in experiments with the Willshaw model. Applying the very efficient CB memory model either in information retrieval systems or as a functional model for reciprocal cortico-cortical pathways requires more than robustness against random noise in the input: Our experiments show also the segmentation ability of CB-retrieval with addresses containing the superposition of pattens, provided even at high memory load.

## 1 INTRODUCTION

From a technical point of view neural associative memories (NAM) provide data storage and retrieval. Neural models naturally imply parallel implementation of storage and retrieval algorithms by the correspondence to synaptic modification and neural activation. With distributed coding of the data the recall in NAM models is fault tolerant: It is robust against noise or superposition in the addresses and against local damage in the synaptic weight matrix. As biological models NAM

have been proposed as general working schemes of networks of pyramidal cells in many places of the cortex.

An important property of a NAM model is its information capacity, measuring how efficient the synaptic weights are used. In the early sixties Steinbuch realized under the name "Lernmatrix" a memory model with binary synapses which is now known as Willshaw model [Ste61, WBLH69]. The great variety of NAM models proposed since then, many triggered by Hopfield's work [Hop82], do not reach the high asymptotic information capacity of the Willshaw model.

For finite network size, the Willshaw model does not optimally retrieve the stored information, since the inner product between matrix colum and input pattern determines the activity for each output neuron independently. For autoassociative pattern completion iterative retrieval can reduce cross talk noise [GM76, GR92, PS92, SSP96]. A simple bidirectional iteration – as in bidirectional associative memory (BAM) [Kos87] – can, however, not improve heteroassociative pattern mapping. For this task we propose CB-retrieval where each retrieval step forms the resulting activity pattern in an autoassociative process that uses the connectivity matrix twice before thresholding, thereby exploiting the stored information more efficiently.

## 2   WILLSHAW MODEL AND CB EXTENSION

Here pattern mapping tasks $x^\nu \to y^\nu$ are considered for a set of *memory patterns*: $\{(x^\nu, y^\nu) : x^\nu \in \{0,1\}^n, y^\nu \in \{0,1\}^m, \nu = 1, ..., M\}$. The number of 1-components in a pattern is called *pattern activity*. The Willshaw model works efficiently, if the memories are *sparse*, i.e., if the memory patterns have the same activities: $|x^\nu| = \sum_{i=1}^n x_i^\nu = a, |y^\nu| = \sum_{i=1}^m y_i^\nu = b \ \forall \ \nu$ with $a << n$ and $b << m$. During learning the set of memory patterns is transformed to the weight matrix by

$$C_{ij} = \min(1, \sum_\nu x_i^\nu y_j^\nu) = \sup_\nu x_i^\nu y_j^\nu.$$

For a given initial pattern $\tilde{x}^\mu$ the retrieval yields the output pattern $\hat{y}^\mu$ by forming in each neuron the dendritic sum $[C\tilde{x}^\mu]_j = \sum_i C_{ij}\tilde{x}_i^\mu$ and by calculating the activity value by threshold comparison

$$\hat{y}_j^\mu = H([C\tilde{x}^\mu]_j - \theta) \ \forall j, \tag{1}$$

with the global threshold value $\theta$ and $H(x)$ denoting the Heaviside function.

For finite sizes and with high memory load, i.e., $0 << P_1 := \text{Prob}\,[C_{ij} = 1]\,(< 0.5)$, the Willshaw model provides no tolerance with respect to errors in the address, see Fig. 1 and 2. A bidirectional iteration of standard *simple retrieval* (1), as proposed in BAM models [Kos87], can therefore be ruled out for further retrieval error reduction [SP97]. In the energy function of the Willshaw BAM

$$E(x, y) = -\sum_{ij} C_{ij} x_i y_j + \Theta' \sum_i x_i + \Theta \sum_j y_j$$

we now indroduce a factor accounting for the magnitudes of dendritic potentials at activated neurons

$$E(x, y) = -\sum_{ij} C_{ij} x_i y_j \frac{a[C^T y]_i + b[Cx]_j}{a + b} + \Theta' \sum_i x_i + \Theta \sum_j y_j. \tag{2}$$

Differentiating the energy function (2) yields the gradient descent equations

$$y_j^{new} = H(\ [Cx]_j^2 + \sum_k \underbrace{\sum_i C_{ij}C_{ik}x_i}_{=:w_{jk}^x}\ y_k - \Theta\ ) \tag{3}$$

$$x_i^{new} = H(\ [C^Ty]_i^2 + \sum_l \underbrace{\sum_j C_{ij}C_{lj}y_j}_{=:w_{il}^y}\ x_l - \Theta'\ ) \tag{4}$$

As new terms in (3) and (4) sums over pattern components weighted with the quantities $w_{jk}^x$ and $w_{il}^y$ occur. $w_{jk}^x$ is the overlap between the matrix columns $j$ and $k$ conditioned by the pattern $x$, which we call a *conditioned link* between $y$-units. Restriction on the conditioned link terms yields a new iterative retrieval scheme which we denote as *crosswise bidirectional (CB) retrieval*

$$y(r+1)_j = H(\sum_{i \in x(r)} C_{ij}[C^Ty(r-1)]_i - \Theta) \tag{5}$$

$$x(r+1)_i = H(\sum_{j \in y(r)} C_{ij}[Cx(r-1)]_j - \Theta') \tag{6}$$

For $r = 0$ pattern $y(r-1)$ has to be replaced by $H([Cx(0)] - \theta)$, for $r > 2$ Boolean ANDing with results from timestep $r - 1$ can be applied which has been shown to improve iterative retrieval in the Willshaw model for autoassociation [SSP96].

## 3  MODEL EVALUATION

Two possible retrieval error types can be distinguished: a *"miss"* error converts a 1-entry in $y^\mu$ to '0' and a *"add"* error does the opposite.

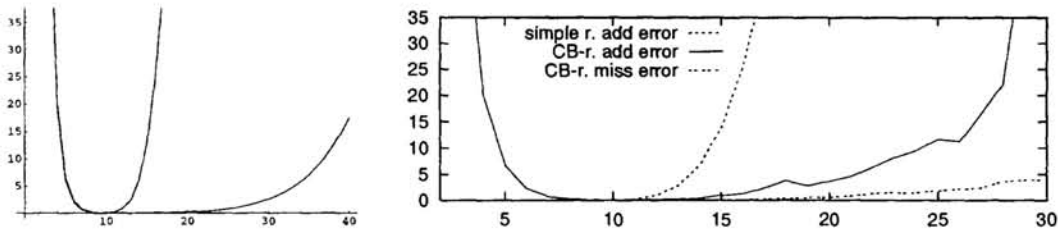

Figure 1: Mean retrieval error rates for $n = 2000$, $M = 15000$, $a = b = 10$ corresponding to a memory load of $P_1 = 0.3$. The $x$-axes display the address activity: $|\tilde{x}^\mu| = 10$ corresponds to a errorfree learning pattern, lower activities are due to miss errors, higher activities due to add errors. Left: Theory – Add errors for simple retrieval, eq. (7) (upper curve) and lower bound for the first step of CB-retrieval, eq. (9). Right: Simulations – Errors for simple and CB retrieval.

The analysis of simple retrieval from the address $\tilde{x}^\mu$ yields with optimal threshold setting $\theta = \tilde{k}$ the add error rate, i.e, the expectation of spurious ones:

$$\hat{\alpha} = (m - b)\text{Prob}\left[r \geq \tilde{k}\right], \tag{7}$$

with the binomial random variable $\text{Prob}\,[r\!=\!l] = B(\lfloor \tilde{x}^\mu \rfloor, P_1)_l$, where $B(n,p)_l :=$ $\binom{n}{l} p^l (1-p)^{n-l}$. $\tilde{\alpha}$ denotes the add error rate and $\bar{k} = |\tilde{x}^\mu| - \tilde{\alpha}$ the number of correct 1-s in the address.

For the first step of CB-retrieval a lower bound of the add error rate $\alpha(1)$ can be derived by the analysis of CB-retrieval with fixed address $x(0) = \tilde{x}^\mu$ and the perfect learning pattern $y^\mu$ as starting patterns in the $y$-layer. In this case the add error rate is:

$$\alpha = (m-b)\text{Prob}\left[r_1 + r_2 \geq \bar{k}b\right],\tag{8}$$

where the random variables $r_1$ and $r_2$ have the distributions:
$\text{Prob}\,[r_1 = l/b] = B(\bar{k}, P_1)_l$ and $\text{Prob}\,[r_2 = l] = B\big(\tilde{\alpha}b, (P_1)^2\big)_l$. Thus,

$$\alpha(1) \geq (m-b)\sum_{s=0}^{\bar{k}} B(\bar{k}, P_1)_s BS\left[\tilde{\alpha}b, (P_1)^2, (\bar{k}-s)b\right],\tag{9}$$

where $BS\,[n,p,t] := \sum_{l=t}^n B(n,p)_l$ is the binomial sum.

In Fig. 1 the analytic results for the first step (7) and (9) can be compared with simulations (left versus right diagram). In the experiments simple retrieval is performed with threshold $\theta = \bar{k}$. CB-retrieval is iterated in the $y$-layer (with fixed address $\tilde{x}$) starting with three randomly chosen 1-s from the simple retrieval result $\hat{y}^\mu$. The iteration is stopped, if a stable pattern at threshold $\Theta = b\bar{k}$ is reached.

The memory capacity can be calculated per pattern component under the assumption that in the memory patterns each component is independent, i.e., the probabilities for a 1 are $p = a/n$ or $q = b/m$ respectively, and the probabilities of an add and a miss error are simply the renormalized rates denoted by $\alpha'$, $\beta'$ and $\tilde{\alpha}'$, $\tilde{\beta}'$ for $x$-patterns and by $\gamma'$, $\delta'$ for $y$-patterns. The information about the stored pattern contained in noisy initial or retrieved patterns is then given by the transinformation $t(p,\alpha',\beta') := i(p) - i(p,\alpha',\beta')$, where $i(p)$ is the Shannon information, and $i(p,\alpha',\beta')$ the conditional information. The heteroassociative mapping is evaluated by the *output capacity:* $A(\tilde{\alpha}',\tilde{\beta}') := Mm\,t(q,\gamma',\delta')/mn$ (in units bit/synapse). It depends on the initial noise since the performance drops with growing initial errors and assumes the maximum, if no fault tolerance is provided, that is, with noiseless initial patterns, see Fig. 2. Autoassociative completion of a distorted $x$-pattern is evaluated by the *completion capacity:* $C(\tilde{\alpha}',\tilde{\beta}') := Mn(t(p,\alpha',\beta')-t(p,\tilde{\alpha}',\tilde{\beta}'))/mn$. A BAM maps and completes at the same time and should be therefore evaluated by the *search capacity* $S := C + A$.

The asymptotic capacity of the Willshaw model is strikingly high: The completion capacity (for autoassociation) is $C^+ = \ln[2]/4$, the mapping capacity (for heteroassociation with input noise) is $A^+ = \ln[2]/2$ bit/syn [Pal91], leading to a value for the search capacity of $(3\ln[2])/4 = 0.52$ bit/syn. To estimate $S$ for general retrieval procedures one can consider a recognition process of stored patterns in the whole space of sparse initial patterns; an initial pattern is "recognized", if it is invariant under a bidirectional retrieval cycle. The so-called *recognition capacity* of this process is an upper bound of the completion capacity and it had been determined as $\ln[2]/2$, see [PS92]. This is achieved again with parameters $M, p, q$ providing $A = \ln[2]/2$ yielding $\ln[2]$ bit/syn as upper bound of the asymptotic search capacity. In summary, we know about the asymptotic search capacity of the CB-model: $0.52 \leq S^+ \leq 0.69$ bit/syn. For experimental results, see Fig. 4.

# 4 EXPERIMENTAL RESULTS

The CB model has been tested in simulations and compared with the Willshaw model (simple retrieval) for addresses with random noise (Fig. 2) and for addresses composed by two learning patterns (Fig. 3). In Fig. 2 the widely enlarged range of high qualtity retrieval in the CB-model is demonstrated for different system sizes.

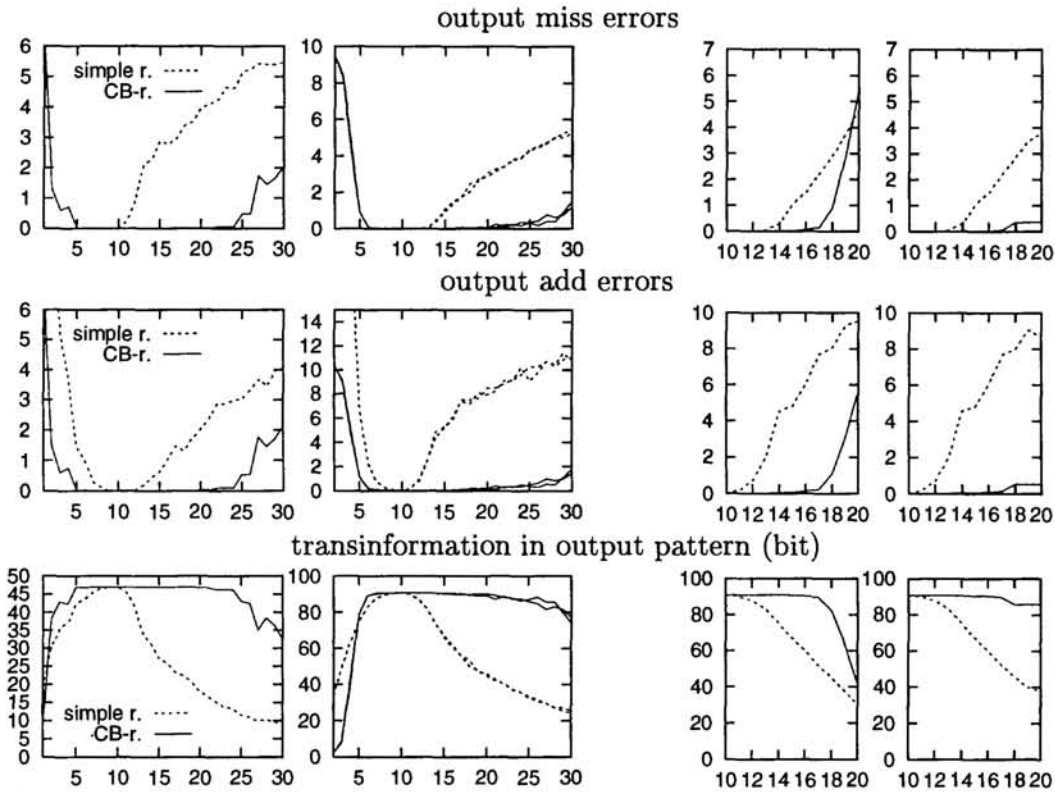

Fig. 2: Retrieval from addresses with random noise. The $x$-axis labeling is as in Fig. 1. Small system with $n = 100$, $M = 35$ (left), system size as in Fig. 1, two trials (right). Output activities adjusted near $|y| = \bar{k}$ by threshold setting.

Fig. 3: Retrieval from addresses composed by two learning patterns. Parameters as in right column of Fig. 2, explanation of left and right column, see text.

In Fig. 3 the address contains one learning pattern and 1-components of a second learning pattern successively added with increasing abscissae. On the right end of each diagram both patterns are completely superimposed. Diagrams in the left column show errors and transinformation, if retrieval results are compared with the learning pattern which is for $|\tilde{x}^\mu| < 20$ dominantly addressed. Simple retrieval errors behave similiar as for random noise in the address (Fig. 2) while the error level of CB-retrieval raises faster if more than 7 adds from the second pattern are present. Diagrams in the right column show the same quantities, if the retrieval result is compared with the closest of the two learning patterns. It can be observed i) that a learning pattern is retrieved even if the address is a complete superposition and ii) if the second pattern is almost complete in the address the retrieved pattern corresponds in some cases to the second pattern. However, in all cases CB-retrieval yields one of the learning pattern pairs and it could be used to generate a good address for further retrieval of the other by deletion of the corresponding 1-components in the original address.

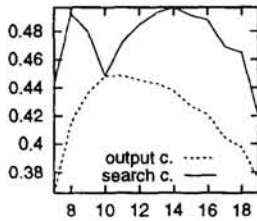

Fig. 4: Output and search capacity of CB retrieval in bit/syn with x-axis labeling as in Fig. 2 for $n = m = 2000$, $a = b = 10$ $M = 20000$. The difference between both curves is the contribution due to $x$-pattern completion, the completion capacity $C$. It is zero for $|x(0)| = 10$, if the initial pattern is errorfree.

The search capacity of the CB model in Fig. 4 is close to the theoretical expectations from Sect. 3, increasing with input noise due to the address completion.

## 5  SPARSE CODING

To apply the proposed NAM model, for instance, in information retrieval, a coding of the data to be accessed into sparse binary patterns is required. A useful extraction of sparse features should take account of statistical data properties and the way the user is acting on them. There is evidence from cognitive psychology that such a coding is typically quite easy to find. The feature encoding, where a person is extracting feature sets to characterize complex situations by a few present features, is one of the three basic classes of cognitive processes defined by Sternberg [Ste77]. Similarities in the data are represented by feature patterns having a large number of present features in common, that is a high overlap: $o(x, x') := \sum_i x_i x'_i$. For text retrieval word fragments used in existing indexing techniques can be directly taken as sparse binary features [Geb87]. For image processing sparse coding strategies [Zet90], and neural models for sparse feature extraction by anti-Hebbian learning [Föl90] have been proposed. Sparse patterns extracted from different data channels in heterogeneous data can simply be concatenated and processed simultaneously in NAM. If parts of the original data should be held in a conventional memory, also these addresses have to be represented by distributed and sparse patterns in order to exploit the high performance of the proposed NAM.

## 6  CONCLUSION

A new bidirectional retrieval method (CB-retrieval) has been presented for the Willshaw neural associative memory model. Our analysis of the first CB-retrieval step indicates a high potential for error reduction and increased input fault tolerance. The asymptotic capacity for bidirectional retrieval in the binary Willshaw matrix has been determined between 0.52 and 0.69 bit/syn. In experiments CB-retrieval showed significantly increased input fault tolerance with respect to the standard model leading to a practical information capacity in the order of the theoretical expectations (0.5 bit/syn). Also the segmentation ability of CB-retrieval with ambiguous addresses has been shown. Even at high memory load such input patterns can be decomposed and corresponding memory entries returned individually. The model improvement does not require sophisticated individual threshold setting [GW95], strategies proposed for BAM like more complex learning procedures, or "dummy augmentation" in the pattern coding [WCM90, LCL95].

The demonstrated performance of the CB-model encourages applications as massively parallel search strategies in Information Retrieval. The sparse coding requirement has been briefly discussed regarding technical strategies and psychological plausibility. Biologically plausible variants of CB-retrieval contribute to more

refined cell assembly theories, see [SWP98].

Acknowledgement: One of the authors (F.T.S.) was supported by grant SO352/3-1 of the Deutsche Forschungsgemeinschaft.

# References

[Föl90]   P. Földiak. Forming sparse representations by local anti-hebbian learning. *Biol. Cybern.*, 64:165–170, 1990.

[Geb87]   F. Gebhardt. Text signatures by superimposed coding of letter triplets and quadruplets. *Information Systems*, 12(2):151–156, 1987.

[GM76]   A.R. Gardner-Medwin. The recall of events through the learning of associations between their parts. *Proceedings of the Royal Society of London B*, 194:375–402, 1976.

[GR92]   W.G. Gibson and J. Robinson. Statistical analysis of the dynamics of a sparse associative memory. *Neural Networks*, 5:645–662, 1992.

[GW95]   B. Graham and D. Willshaw. Improving recall from an associative memory. *Biological Cybernetics*, 72:337–346, 1995.

[Hop82]   J.J. Hopfield. Neural networks and physical systems with emergent collective computational abilities. *Proceedings of the National Academy of Sciences, USA*, 79, 1982.

[Kos87]   B. Kosko. Adaptive bidirectional associative memories. *Applied Optics*, 26(23):4947–4971, 1987.

[LCL95]   C.-S. Leung, L.-W. Chan, and E. Lai. Stability, capacity and statistical dynamics of second-order bidirectional associative memory. *IEEE Trans. Syst, Man Cybern.*, 25(10):1414–1424, 1995.

[Pal91]   G. Palm. Memory Capacities of Local Rules for Synaptic Modification. *Concepts in Neuroscience*, 2:97–128, 1991.

[PS92]   G. Palm and F. T. Sommer. Information capacity in recurrent McCulloch-Pitts networks with sparsely coded memory states. *Network*, 3:1–10, 1992.

[SP97]   F. T. Sommer and G. Palm. Improved bidirectional retrieval of sparse patterns stored by Hebbian learning. *Submitted to Neural Networks*, 1997.

[SSP96]   F. Schwenker, F. T. Sommer, and G. Palm. Iterative retrieval of sparsely coded associative memory patterns. *Neural Networks*, 9(3):445 – 455, 1996.

[Ste61]   K. Steinbuch. Die Lernmatrix. *Kybernetik*, 1:36–45, 1961.

[Ste77]   R. J. Sternberg. *Intelligence, information processing and analogical reasoning.* Hillsdale, NJ, 1977.

[SWP98]   F. T. Sommer, T. Wennekers, and G. Palm. Bidirectional completion of Cell Assemblies in the cortex. In *Computational Neuroscience: Trends in Research.* Plenum Press, 1998.

[WBLH69]   D. J. Willshaw, O. P. Buneman, and H. C. Longuet-Higgins. Nonholographic associative memory. *Nature*, 222:960–962, 1969.

[WCM90]   Y. F. Wang, J. B. Cruz, and J. H. Mulligan. Two coding stragegies for bidirectional associative memory. *IEEE Trans. Neural Networks*, 1(1):81–92, 1990.

[Zet90]   C. Zetsche. Sparse coding: the link between low level vision and associative memory. In R. Eckmiller, G. Hartmann, and G. Hauske, editors, *Parallel Processing in Neural Systems and Computers.* Elsevier Science Publishers B. V. (North Holland), 1990.
